# Phonetic Classification and Recognition Using the Multi-Layer Perceptron

**Hong C. Leung, James R. Glass,**
**Michael S. Phillips, and Victor W. Zue**
Spoken Language Systems Group
Laboratory for Computer Science
Massachusetts Institute of Technology
Cambridge, Massachusetts 02139, U.S.A.

## Abstract

In this paper, we will describe several extensions to our earlier work, utilizing a segment-based approach. We will formulate our segmental framework and report our study on the use of multi-layer perceptrons for detection and classification of phonemes. We will also examine the outputs of the network, and compare the network performance with other classifiers. Our investigation is performed within a set of experiments that attempts to recognize 38 vowels and consonants in American English independent of speaker. When evaluated on the TIMIT database, our system achieves an accuracy of 56%.

## 1  Introduction

Thus far, the neural networks research community has placed heavy emphasis on the problem of pattern classification. In many applications, including speech recognition, one must also address the issue of *detection*. Thus, for example, one must detect the presence of phonetic segments as well as classify them. Recently, the community has moved more towards recognition of continuous speech. A network is typically used to label every frame of speech in a frame-based recognition system [Franzini 90, Morgan 90, Tebelskis 90].

Our goal is to study and exploit the capability of ANN for speech recognition, based on the premise that ANN may offer a flexible framework for us to utilize our

improved, albeit incomplete, speech knowledge. As an intermediate milestone, this paper extends our earlier work on phonetic classification to context-independent phonetic recognition. Thus we need to locate as well as identify the phonetic units. Our system differs from the majority of approaches in that a segmental framework is adopted. The network is used in conjunction with acoustic segmentation procedures to provide a phonetic string for the entire utterance.

## 2  Segmental Formulation

In our segmental framework, a phonetic unit is mapped to a segment explicitly delineated by a begin and end time in the speech signal. This is motivated by the belief that a segmental framework offers us more flexibility in applying our speech knowledge than is afforded by a frame-based approach. As a result, a segment-based approach could ultimately lead to superior modelling of the temporal variations in the realization of underlying phonological units.

Let $\hat{\alpha}$ denote the best sequence of phonetic units in an utterance. To simplify the problem, we assume that $p(s_i) = p(s_i | \alpha_j)$, where $s_i$ stand for the $i^{\text{th}}$ time segment that has *one and only one* phoneme in it, and $\alpha_j$ stands for the best phoneme label in $s_i$. Thus the probability of the best sequence, $p(\hat{\alpha})$, is:

$$p(\hat{\alpha}) = \prod_{s_i \in \vec{s}} p(\alpha_j) p(s_i); \qquad 1 \le j \le N \qquad (1)$$

where $\vec{s}$ is any possible sequence of time segments consisting of $\{s_1, s_2, ...\}$, $p(s_i)$ is the probability of a valid time segment, and $N$ is the number of possible phonetic units. In order to perform recognition, the two probabilities in Equation 1 must be estimated. The first term, $p(\alpha_j)$, is a set of phoneme probabilities and thus can be viewed as a classification problem. The second term, $p(s_i)$, is a set of probabilities of valid time regions and thus can be estimated as a segmentation problem.

### 2.1  Segmentation

In order to estimate the segment probabilities, $p(s_i)$, in Equation 1, we have formulated segmentation into a boundary classification problem. Let $b_l$ and $b_r$ be the left and right boundary of a time segment, $s_i$, respectively, as shown in Figure 1a. Let $\{b_1, b_2, .., b_K\}$ be the set of boundaries that might exist within $s_i$. These boundaries can be proposed by a boundary detector, or they can simply occur at every frame of speech. We define $p(s_i)$ to be the joint probability that the left and right boundaries *exist* and all other boundaries within $s_i$ *do not* exist. To reduce the complexity of the problem, assume $b_j$ is statistically independent of $b_k$ for $\forall j \neq k$. Thus,

$$\begin{aligned} p(s_i) &= p(b_l, \bar{b}_1, \bar{b}_2, .., \bar{b}_K, b_r) \\ &= p(b_l) p(\bar{b}_1) p(\bar{b}_2) ... p(\bar{b}_K) p(b_r), \end{aligned} \qquad (2)$$

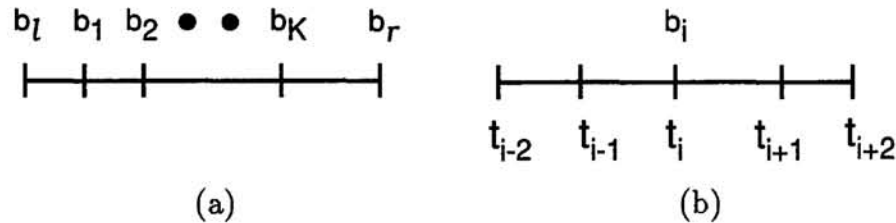

Figure 1: Schematic diagrams for estimation of (a) segment probability, $p(s_i)$, and (b) boundary probability, $p(b_k)$. The boundaries can be proposed by a boundary detector, or they can simply occur at every frame. See text.

where $p(b_l)$ and $p(b_r)$ stand for the probability that the left and right boundary exist, respectively, $p(b_k)$ stands for the probability that the $k^{\text{th}}$ boundary does not exist. As a result, the probability of a segment, $p(s_i)$ can be obtained by computing the probabilities of the boundaries, $p(b_k)$, subsumed by the segment. As we will discuss in a later section, by using the time-aligned transcription, we can train the boundary probabilities in a supervised manner.

## 2.2    Phonetic Classification

Once the probability of a segment, $p(s_i)$, is obtained, we still need to classify it, i.e. compute the probabilities of the phonetic units in the segment, $p(\alpha_j)$. Again, the time-aligned transcription can be used to train the probabilities in a supervised manner. We have discussed this in earlier papers [Leung 89, Leung 90]. In a later section, we will discuss some of our recent experimental results.

# 3    Experiments

## 3.1    Tasks and Corpora

The experiments described in this paper deal with classification and recognition of 38 phonetic labels representing 14 vowels, 3 semivowels, 3 nasals, 8 fricatives, 2 affricates, 6 stops, 1 flap and 1 silence. Within the context of classification, the networks are given a segment of the speech signal, and are asked to determine its phonetic identity. Within the context of recognition, the networks are given an utterance, and are asked to determine the identity and locations of the phonetic units in the utterance. All experiments were based on the sentences in the TIMIT database [Lamel 86]. As summarized in Table 1, Corpus I contains 1,750 sx sentences spoken by 350 male and female speakers, resulting in a total of 64,000 phonetic tokens. Corpus II contains 4,400 sx and si sentences spoken by 550 male and female speakers, resulting in a total of 165,000 phonetic tokens.

## 3.2    Phonetic Classification

As previously discussed, estimation of the probability, $p(\alpha_j)$ in Equation 1 can be viewed as a classification problem. Many statistical classifiers can be used. We have

| Corpus | Set | Speakers | Sentences | Tokens | Type |
|--------|-----|----------|-----------|--------|------|
| I | training | 300 | 1500 | 55,000 | sx |
|   | testing | 50 | 250 | 9,000 | sx |
| II | training | 500 | 4000 | 150,000 | sx/si |
|    | testing | 50 | 400 | 15,000 | sx/si |

Table 1: Corpora I and II extracted from the TIMIT database. Corpus I contains only sx sentences, whereas Corpus II contains both sx and si sentences. The speakers in the testing sets for both Corpus I and Corpus II are the same.

chosen to use the MLP, due to its discriminatory capability, as well as its flexibility in that it does not make assumptions about specific statistical distributions or distance metrics. In addition, earlier work shows that the outputs of MLP can approximate posteriori probabilities [Bourlard 88]. To train the network, we adopt procedures such as center initialization, input normalization, adaptive gain, and modular training [Leung 90]. The input representation was identical to that in the SUMMIT system, and consisted of 82 acoustic attributes [Zue 89]. These segmental attributes were generated automatically by a search procedure that uses the training data to determine the settings of the free parameters of a set of generic property detectors using an optimization procedure [Phillips 88].

### 3.3 Boundary Classification

In our segmental framework formulated in Equation 1, the main difference between classification and recognition is the incorporation of a probability for each segment, $p(s_i)$. As described previously in Equation 2, we have simplified the problem of estimating $p(s_i)$ to one of determining the probability that a boundary exists, $p(b_k)$.

To estimate $p(b_k)$, a MLP with two output units is used, one for the valid boundaries and the other for the extraneous boundaries. By referencing the time-aligned phonetic transcription, the desired outputs of the network can be determined. In our current implementation $p(b_k)$ is determined using four abutting segments, as shown in Figure 1b. These segments are proposed by the boundary detector in the SUMMIT system. Let $t_i$ stand for the time at which $b_i$ is located, and $s_i$ stand for the segment between $t_i$ and $t_{i+1}$, where $t_{i+1} > t_i$. The boundary probability, $p(b_i)$, is then determined by using the average mean-rate response [Seneff 88] in $s_{i-2}, s_{i-1}, s_i$, and $s_{i+1}$ as inputs to the MLP. Thus the network has altogether 160 input units.

### 3.4 Results

#### 3.4.1 Phonetic Classification

In the phonetic classification experiments, the system classified a token extracted from a phonetic transcription that had been aligned with the speech waveform. Since there was no detection involved in these experiments only substitution errors were possible.

| | Classifier | Correct | Parameters |
|---|---|---|---|
| I | SUMMIT | 70% | 2,200 |
| I | Gaussian | 70% | 128,000 |
| I | MLP | 74% | 15,000 |
| II | MLP | 76% | 30,000 |

Table 2: Phonetic classification results using the SUMMIT classifier, Gaussian classifier, and MLP. Also shown are the number of parameters in the classifiers.

In the first set of experiments, we compared results based on Corpus I, using different classifiers. As Table 2 shows, the baseline speaker-independent classification performance of SUMMIT on the testing data was 70%. When Gaussian classifiers with full covariance matrices were used, we found that the performance is also about 70%. Finally, when the MLP is used, a performance of 74% is achieved.

Although the sx sentences were designed to be phonetically balanced, the 1,750 sentences in Corpus I are not distinct. In the second set of experiments, we evaluated the MLP classifier on Corpus II, which include both the sx and si sentences.[1] As shown in Table 2, the classifier achieves 76%.

**Parameters:** The networks used as described in Table 2 have only 1 hidden layer. The number of hidden units in the network can be 128 or 256, resulting in 15,000 or 30,000 connections. For comparison, Table 2 also shows the number of parameters for the SUMMIT and Gaussian classifiers. While the SUMMIT classifier requires only about 2,200 parameters, the Gaussian classifiers require as much as 128,000 parameters, an order of magnitude more than the MLP. These numbers also give us some idea about the computational requirements for different classifiers, since the required number of multiplications is about the same as the number of parameters.

**Network Outputs:** We have chosen the network to estimate the phoneme probabilities. When the network is trained, the target values are either 1 or 0. However, if the network is over-trained, its output values may approach either 1 or 0, resulting in poor estimates of the posterior probabilities. Figure 2 shows two distributions for the output values of the network for 3600 tokens from the test set. Figure 2a corresponds to the ratio of the highest output value to the sum of the network output values, whereas Figure 2b corresponds to the second highest output value. We can see that both distributions are quite broad, suggesting that the network often makes "soft" decisions about the phoneme labels. We feel that this is important since in speech recognition, we often need to combine scores or probabilities from different parts of the system.

### 3.4.2  Boundary Classification

We have evaluated the boundary classifier using the training and testing data in Corpus I. By using 32 hidden units, the network can classify 87% of the boundaries in the test set correctly.

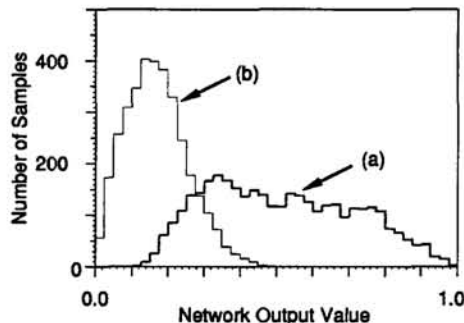

Figure 2: Histograms for the output values of the network extracted from 3600 samples: (a) the highest output values, and (b) the second highest output values.

| Corpus | Classifer | Segment | Accuracy |
|--------|-----------|---------|----------|
| I | Baseline | Binary Hierarchy | 47% |
| I | MLP | Binary Hierarchy | 50% |
| I | MLP | Stochastic Pruning | 54% |
| II | MLP | Stochastic Pruning | 56% |

Table 3: Phonetic recognition results using binary hierarchy (dendrogram), and boundary pruning. No duration, bigram, or trigram statistics have been used. Errors include substitutions, deletions, and insertions.

### 3.4.3  Phonetic Recognition

One of the disadvantages of our segmental framework is that the amount of computation involved can be very significant, since a segment can begin and end at any frame of an utterance. We have explored various pruning strategies. In this paper, we will report our results using stochastic pruning and binary hierarchy [Leung 90a]. We have found that such pruning strategies can reduce the amount of computation by about 3 orders of magnitude.

The results of the phonetic recognition experiments are shown in Table 3. No duration, bigram, or trigram statistics have been used. The baseline performance of the current SUMMIT system on Corpus I is 47%, including substitution, deletion, and insertion errors. When the MLP was used in place of the classifier in the current SUMMIT system using also the binary hierarchical representation, the performance improved to 50%. When the MLP was used with stochastic pruning technique, the performance improved to 54%. Finally, by using the network trained and tested on Corpus II, the performance improved to 56%.

## 4    Discussion

In summary, we have discussed a segmental approach for phonetic recognition. We have also examined the outputs of the network, and compared performance results and computational requirements with different classifiers. We have shown that decisions made by the network are quite "soft", and that the network yields results favorable to other more traditional classifiers. Future work includes the use of context-dependent models for phonetic and boundary classification, utilization of other phonological units, and extension to recognition of continuous speech.

## Footnotes

[1]All the si sentences in TIMIT are distinct.

## References

[Bourlard 88] Bourlard, H., and C.J. Wellekens, "Links between Markov Models and Multilayer Perceptrons," *Advances in Neural Information Processing Systems 1*, Morgan Kaufmann, 1988.

[Franzini 90] Franzini, M.A., K.F. Lee, and A. Waibel, "Connectionist Viterbi Training: A New Hybrid Method for Continuous Speech Recognition," *Proc. ICASSP-90*, Albuquerque, NM, USA, 1990.

[Lamel 86] Lamel, L.F., R.H. Kassel, and S. Seneff, "Speech Database Development: Design and Analysis of the Acoustic Phonetic Corpus," *Proc. DARPA Speech Recognition Workshop*, 1986.

[Leung 89] Leung, H.C., *The Use of Artificial Neural Networks of Phonetic Recognition*, Ph.D. Thesis, Mass. Inst. of Tech., 1989.

[Leung 90] Leung, H.C., and V.W. Zue, "Phonetic Classification Using Multi-Layer Perceptrons," *Proc. ICASSP-90*, Albuquerque, 1990.

[Leung 90a] Leung, H., Glass, J., Phillips, M., and Zue, V., "Detection and Classification of Phonemes Using Context-independent Error Back-Propagation," *Proc. International Conference on Spoken Language Processing*, Kobe, Japan, 1990.

[Morgan 90] Morgan, N., and H. Bourlard, "Continuous Speech Recognition Using Multilayer Perceptrons with Hidden Markov Models," *Proc. ICASSP-90*, Albuquerque, NM, USA, 1990.

[Phillips 88] Phillips, M.S., "Automatic Discovery of Acoustic Measurements for Acoustic Classification," *J. Acoust. Soc. Amer.*, Vol. 84, 1988.

[Seneff 88] Seneff, S. "A Joint Synchrony/Mean-Rate Model of Auditory Speech Processing," *Proc. J. of Phonetics*, 1988.

[Tebelskis] Tebelskis, J., and A. Waibel, "Large Vocabulary Recognition Using Linked Predictive Neural Networks," Proc. ICASSP-90, Albuquerque, NM, USA, 1990.

[Zue 89] Zue, V., J. Glass, M. Phillips, and S. Seneff, "The MIT SUMMIT Speech Recognition System: A Progress Report," *Proceedings of DARPA Speech and Natural Language Workshop*, February, 1989.
